# Asymptotics of Alpha-Divergence Variational Inference Algorithms with Exponential Families

**François Bertholom**
SAMOVAR, Télécom Sud-Paris
Institut Polytechnique de Paris, France
francois.bertholom@telecom-sudparis.eu

**Randal Douc**
SAMOVAR, Télécom Sud-Paris
Institut Polytechnique de Paris, France
randal.douc@telecom-sudparis.eu

**François Roueff**
LTCI, Télécom Paris
Institut Polytechnique de Paris, France
francois.roueff@telecom-paris.fr

## Abstract

Recent works in Variational Inference have examined alternative criteria to the commonly used exclusive Kullback-Leibler divergence. Encouraging empirical results have been obtained with the family of alpha-divergences, but few works have focused on the asymptotic properties of the proposed algorithms, especially as the number of iterations goes to infinity. In this paper, we study a procedure that ensures a monotonic decrease in the alpha-divergence. We provide sufficient conditions to guarantee its convergence to a local minimizer of the alpha-divergence at a geometric rate when the variational family belongs to the class of exponential models. The sample-based version of this ideal procedure involves biased gradient estimators, thus hindering any theoretical study. We propose an alternative unbiased algorithm, we prove its almost sure convergence to a local minimizer of the alpha-divergence, and a law of the iterated logarithm. Our results are exemplified with toy and real-data experiments.

## 1 Introduction

Many statistical inference problems involve computing or sampling from intractable probability densities. Variational Inference [3, 19] is a broad class of methods that turn this kind of inference problems into optimization programs. Given a family of tractable densities $\mathcal{Q}$, the goal is to find, within that family, a fine approximation $q$ of the targeted density $p$ according to a given criterion. The discrepancy between $p$ and $q$ is usually measured by the exclusive Kullback-Leibler divergence $D_{\mathrm{KL}}(q \,\|\, p)$, for it is quite convenient to use from a computational standpoint. However, it may lead to approximations that tend to underestimate the variance of the target [27], a behavior known as zero-forcing, or mode-seeking when the variational density is unimodal. This can be highly detrimental to the quality of the posterior approximation, for instance when the target is multimodal. To overcome this difficulty, recent research has experimented with other divergence measures, such as the inclusive KL divergence $D_{\mathrm{KL}}(p \,\|\, q)$ [28], or the more general alpha-divergence family [1], for instance in [9, 15, 25]. Specifically, in [9], the authors introduced a family of iterative algorithms that perform *monotonic* alpha-divergence minimization, meaning that the criterion is guaranteed to decrease at each new iteration. Although numerical experiments suggested a strong potential for their method, the convergence properties of the proposed algorithm remained to be determined.

After briefly giving some background on alpha-divergence Variational Inference (Section 2), we study the convergence properties of the monotonic alpha-divergence minimization algorithm introduced in

[9], assuming that the variational family $\mathcal{Q}$ is an exponential family of densities (Section 3). We give sufficient conditions for the algorithm to converge toward a local minimizer of the alpha-divergence, and show that the rate of convergence is asymptotically geometric. In Section 4, we explain why the empirical counterpart of this algorithm will not converge to minimizers of the alpha-divergence. Instead, we propose an alternative algorithm that has more advantageous properties. We prove its almost sure convergence, as well as a law of the iterated logarithm. Section 5 is dedicated to extending our algorithm to the training of Variational Auto-Encoders [22]. We exhibit links to variational bounds that are known and have already been studied in the literature [7, 25]. While the main focus of this paper is theoretical, we also provide empirical results on both toy examples and real data in Section 6.

**Related work**    Alpha-divergence minimization has lately become a topic of interest in Variational Inference [5, 7, 8, 9, 10, 26, 35, 38], and most of the currently existing algorithms that perform this task rely on biased gradient estimators [15, 25, 31]. While these strategies have yielded promising empirical results, they have only superficially been studied in theory. In [7], the authors provide guarantees for methods based on the variational bound introduced in [25], corroborating empirical findings in [14]. The convergence of other Variational Inference algorithms has been studied. In [28, 39], the authors use Markov Chain Monte-Carlo to optimize the inclusive Kullback-Leibler divergence, and provide conditions for the almost-sure convergence of their algorithm. Convergence guarantees and rates for black-box variational inference with Gaussian variational families have been obtained in [12]. Even though we use different proof techniques, our framework can be seen as an instantiation of Minorize-Maximization algorithms [18, 23], for which convergence guarantees and rates can be obtained, often at the cost of smoothness assumptions [24].

## 2   Monotonic alpha-divergence minimization with exponential families

**Variational Inference.**    Let $(\mathsf{Y}, \mathcal{Y}, \nu)$ and $(\mathsf{X}, \mathcal{X}, \nu')$ be sigma-finite measure spaces, and suppose that we are given some data $\boldsymbol{x} = (x_1, \ldots, x_n)$ in $\mathsf{X}$. We make the assumption that it stems from a latent variable probabilistic model $p_\theta(\boldsymbol{x}, y) = p_0(y)p_\theta(\boldsymbol{x} \,|\, y)$, where $\theta$ is some parameter and $y \in \mathsf{Y}$ is a latent variable. The prior $p_0(y)$ and the conditional likelihood $p_\theta(\boldsymbol{x} \,|\, y)$ are specified. We are interested in estimating the posterior density $p_\theta(y \,|\, \boldsymbol{x}) = p_\theta(y, \boldsymbol{x})/p_\theta(\boldsymbol{x})$, which is intractable in general due to the normalizing constant $p_\theta(\boldsymbol{x})$. Variational Inference methods aim at finding an approximation of the posterior within a family of tractable probability densities $\mathcal{Q} = \{q_\eta(\cdot), \, \eta \in E\}$, where $E \subseteq \mathbb{R}^d$ is some subset of parameters. The quality of the resulting approximation is estimated by a given criterion that quantifies the dissimilarity between the variational estimate $q_\eta(\cdot)$ and the target $p_\theta(\cdot \,|\, \boldsymbol{x})$. We will work with the alpha-divergence [1, 6], i.e., we will try to minimize

$$D_\alpha\big(q_\eta(\cdot) \,\|\, p_\theta(\cdot \,|\, \boldsymbol{x})\big) = \frac{1}{\alpha(\alpha - 1)} \int_{\mathsf{Y}} \left[ \left( \frac{q_\eta(y)}{p_\theta(y \,|\, \boldsymbol{x})} \right)^\alpha - 1 \right] p_\theta(y \,|\, \boldsymbol{x})\nu(\mathrm{d}y).$$

The parameter $\alpha$ allows tuning between mass-covering or zero-forcing behaviors. Precisely, the case $\alpha \longrightarrow 0$ corresponds to the inclusive KL divergence, while $\alpha \longrightarrow 1$ recovers the exclusive KL, and $\alpha = 0.5$ is Hellinger's distance. The alpha-divergence is defined similarly to the Rényi divergence [25, 32, 36], which writes $D_\alpha^R(q_\eta \,\|\, p_\theta) = \frac{1}{\alpha - 1} \log \int q_\eta^\alpha p_\theta^{1-\alpha} \mathrm{d}\nu$ for all $\alpha \neq 1$. It is then equivalent to minimize either of these two objectives. In the remainder of this paper, we set $\alpha \in (0, 1)$.

Oftentimes, we only know $p_\theta(y \,|\, \boldsymbol{x})$ up to a positive constant, say $p_\theta(y \,|\, \boldsymbol{x}) \propto p(y)$. Most of the time, the easiest choice is $p = p_\theta(\boldsymbol{x}, \cdot)$. Since $\boldsymbol{x}$ remains fixed throughout the optimization process, we drop the dependency in $\boldsymbol{x}$ and simply write $p(y)$. Minimizing the alpha-divergence amounts to solving $\inf_{\eta \in E} \mathcal{L}_\alpha(\eta)$, where

$$\mathcal{L}_\alpha(\eta) = \frac{1}{\alpha(\alpha - 1)} \int_{\mathsf{Y}} q_\eta(y)^\alpha p(y)^{1-\alpha} \nu(\mathrm{d}y). \tag{1}$$

Since $\int_{\mathsf{Y}} p \, \mathrm{d}\nu < +\infty$, Hölder's inequality ensures that this quantity is well-defined for all $\eta \in E$.

**Useful definitions and notation.**    For readability, we define and use throughout the paper

$$\varphi_\eta^\alpha(y) = q_\eta(y)^\alpha p(y)^{1-\alpha}, \qquad \ell_\alpha(\eta) = \int_{\mathsf{Y}} \varphi_\eta^\alpha(y)\nu(\mathrm{d}y), \qquad \check{\varphi}_\eta^\alpha = \varphi_\eta^\alpha / \ell_\alpha(\eta). \tag{2}$$

Note that $\check{\varphi}_\eta^\alpha$ is a probability density function (p.d.f.) with respect to $\nu$. Subscripts to $\mathbb{E}$ or $\mathrm{Cov}$ indicate the integrating probability measure, e.g., for a p.d.f. $f$ and a measurable function $g$, $\mathbb{E}_f[g] = \int_{\mathsf{Y}} g(y) f(y) \nu(\mathrm{d}y)$. When $\eta_\bullet$ has a subscript $\bullet$ and acts as a subscript, we only keep $\bullet$, e.g., $\varphi_\star^\alpha$ may be used in place of $\varphi_{\eta_\star}^\alpha$.

**Monotonic alpha-divergence minimization.** In [9], the authors propose an iterative algorithm to perform alpha-divergence minimization in a Variational Inference framework. Given an initial parameter $\eta_0 \in E$ and non-negative numbers $(b_t)_{t \geq 0}$, we construct a sequence $(\eta_t)_{t \geq 0}$ using

$$\eta_{t+1} = \arg\max_{\eta \in E} \int_{\mathsf{Y}} \left[ q_{\eta_t}(y)^\alpha p(y)^{1-\alpha} + b_t q_{\eta_t}(y) \right] \log\left( \frac{q_\eta(y)}{q_{\eta_t}(y)} \right) \nu(\mathrm{d}y), \tag{3}$$

assuming that the $\arg\max$ is uniquely defined at each iteration (in this paper, this condition will always be satisfied). Any value of $\eta_{t+1}$ that makes the function in the $\arg\max$ negative ensures a strict decrease in the alpha-divergence between the variational distribution and the target [9, Theorem 1]. Since this function is null at $\eta_t$, each new iterate is at least as good as the previous one. The point of taking the $\arg\max$ is to obtain explicit update formulas when the variational family is well-chosen. However, when the search space is constrained, these explicit formulas may yield iterates outside the feasible set. To overcome this difficulty, the regularization term in $b_t$ is meant to keep the iterates within acceptable bounds. Therfore, it can be understood as an inverse step-size.

**The case of the exponential family.** We respectively denote by $\langle \cdot, \cdot \rangle$ and $\| \cdot \|$ the canonical inner product on $\mathbb{R}^d$ and the induced norm. Assume that the variational family $\mathcal{Q}$ belongs to the class of canonical exponential models, i.e., it is a family of probability densities that write, for all $\eta$ in the feasible set $E := \{\eta \in \mathbb{R}^d, |A(\eta)| < +\infty\}$ and all $y \in \mathsf{Y}$,

$$q_\eta(y) = \kappa(y) \exp\left[ \langle \eta, S(y) \rangle - A(\eta) \right], \tag{4}$$

where $\kappa : \mathsf{Y} \to \mathbb{R}_+$ is a function, $S : \mathsf{Y} \to E$ is a sufficient statistic for the natural parameter $\eta$, and $A : E \to \mathbb{R}$ is the log-partition function. We will assume that $\kappa > 0$ $\nu$-almost everywhere, and that the family $\mathcal{Q}$ is minimal, i.e., the components of $S(y)$ are not linearly constrained for $\nu$-almost all $y \in \mathsf{Y}$. Finally, we suppose that $\mathcal{Q}$ is regular, that is, the feasible set $E$ is open. These assumptions about $\mathcal{Q}$ are later referred to as condition (H1). An important property of such families is that for all $\eta \in E$, we have $\partial_\eta A(\eta) = \mathbb{E}_{q_\eta}[S]$, where $\partial_\eta A$ is the gradient of $A$ with respect to $\eta$. Unless otherwise specified, $\partial A$ will shorthand $\partial_\eta A$. The gradient $\partial A$ is a $\mathcal{C}^\infty$-diffeomorphism from $E$ to $F := \partial A(E)$. Let us define

$$\mathcal{R}(\eta) := \mathbb{E}_{\check{\varphi}_\eta^\alpha}[S] = \frac{1}{\ell_\alpha(\eta)} \int_{\mathsf{Y}} S(y) q_\eta(y)^\alpha p(y)^{1-\alpha} \nu(\mathrm{d}y). \tag{5}$$

It is shown in [9, Theorem 4] that, given $\eta_0 \in E$ and $\mu_0 = \mathbb{E}_{q_{\eta_0}}[S]$, algorithm (3) is equivalent to

$$\begin{cases} \mu_{t+1} &= \gamma_t \mathcal{R}(\eta_t) + (1 - \gamma_t)\mu_t \\ \eta_{t+1} &= (\partial A)^{-1}(\mu_{t+1}) \end{cases} \tag{6}$$

where $\gamma_t = \nu(\varphi_t^\alpha)/(\nu(\varphi_t^\alpha) + b_t)$ belongs to $(0, 1)$ when $b_t \in (0, +\infty)$. It is guaranteed by [9, Theorem 4] that the $\arg\max$ in (3) is well-defined if and only if $\mu_{t+1}$ belongs to $F$. Since $F$ is an open convex set (c.f. [37, Theorem 3.3]), it is always possible to find a valid $\gamma_t \in (0, 1)$.

**Empirical algorithms.** The recursion (6) involves the expectation $\mathbb{E}_{\check{\varphi}_\eta^\alpha}[S]$, which does not admit a closed form in general and thus needs to be computed *via* Monte-Carlo methods [34]. Assume that for any $\eta \in E$, we can easily generate $K$ independent and identically distributed (i.i.d.) samples $y_1, \ldots, y_K \sim q_\eta(\cdot)$. In the empirical setting, the mean parameterization of exponential families is better suited to theoretical analyses. It is defined by $\mu = \partial A(\eta)$, and, using this parameterization, we introduce

$$\hat{\mathcal{E}}(\mu; K) = \frac{1}{K} \sum_{i=1}^{K} S(y_i) \left( \frac{p(y_i)}{q_\eta(y_i)} \right)^{1-\alpha} \qquad \text{and} \qquad \mathcal{E}(\mu) = \mathbb{E}\left[ \hat{\mathcal{E}}(\mu; 1) \right], \tag{7}$$

$$\hat{\ell}_\alpha^*(\mu; K) = \frac{1}{K} \sum_{i=1}^{K} \left( \frac{p(y_i)}{q_\eta(y_i)} \right)^{1-\alpha} \qquad \text{and} \qquad \ell_\alpha^*(\mu) = \mathbb{E}\left[ \hat{\ell}_\alpha^*(\mu; 1) \right]. \tag{8}$$

In what follows, when $\eta_\bullet$ has a subscript $\bullet$, we will denote by $\mu_\bullet$ the associated mean.

In the iterative empirical framework, it is implicitly assumed that, given all the random variables generated up to step $t$, the $(y_i)_{1 \le i \le K}$ used in the above definitions of $\hat{\mathcal{E}}(\mu_t; K)$ and $\hat{\ell}_\alpha^*(\mu_t; K)$ only depend on $\mu_t$.

As will be clear from Section 3, the recursion (6) can be seen as an iterative algorithm converging to a point $\eta_\star$ that satisfies $\mathcal{R}(\eta_\star) = \mu_\star$. Since $\mathcal{R}(\eta) = \mathcal{E}(\mu)/\ell_\alpha^*(\mu)$, we naturally replace (6) by the (computable) empirical version. Dropping the dependency in $K$ for notational convenience, we get

$$\mu_{t+1} = \mu_t + \gamma_t \left[ \frac{\hat{\mathcal{E}}(\mu_t)}{\hat{\ell}_\alpha^*(\mu_t)} - \mu_t \right]. \tag{9}$$

This is a special case of Robbins-Monro algorithms [33]. To find the zeros of some deterministic function $h : E \to E$, such procedures consist in running the recursion

$$\mu_{t+1} = \mu_t + \gamma_t \left[ h(\mu_t) + r_{t+1} \right], \tag{10}$$

where $(\gamma_t)$ is a decreasing positive sequence, and $(r_t)$ are vector-valued random variables with null conditional means given the previous iterations. The issue with (9) is that $\hat{\mathcal{E}}(\mu_t)/\hat{\ell}_\alpha^*(\mu_t)$ is a biased estimator of $\mathcal{E}(\mu_t)/\ell_\alpha^*(\mu_t)$, hence it can only converge to a zero of $h : \mu \mapsto \mathbb{E}\left[ \hat{\mathcal{E}}(\mu)/\hat{\ell}_\alpha^*(\mu) \right] - \mu$, not to a limit point of (6). We propose an alternative procedure defined by the recursion

$$\mu_{t+1} = \mu_t + \gamma_t \left[ \hat{\mathcal{E}}(\mu_t) - \mu_t \hat{\ell}_\alpha^*(\mu_t) \right]. \tag{11}$$

In Section 4.1, we justify the choice of this algorithm, providing a variety of arguments.

## 3 Convergence rates for the monotonic algorithm

In this section, we give and discuss conditions for the convergence of the monotonic $\alpha$-divergence minimization algorithm (6) when the integral $\mathcal{R}(\eta)$ is evaluated exactly. We start by giving some definitions. For all $\gamma \in [0, 1]$, we define the mapping

$$\mathcal{M}_\gamma : \eta \mapsto (\partial A)^{-1}\left[ \gamma \mathcal{R}(\eta) + (1 - \gamma)\partial A(\eta) \right], \tag{12}$$

so that (6) can be written $\eta_{t+1} = \mathcal{M}_{\gamma_t}(\eta_t)$. Observe that $\mathcal{M}_\gamma$ is defined on the set

$$E_\gamma = \left\{ \eta \in E, \ \gamma \mathcal{R}(\eta) + (1 - \gamma)\partial A(\eta) \in F \right\}. \tag{13}$$

We say that $\eta$ is a fixed point of $\mathcal{M}_\gamma$ and we write $\eta \in \text{Fix}(\mathcal{M}_\gamma)$ if and only if $\mathcal{M}_\gamma(\eta) = \eta$. Interestingly, these fixed points are characterized by a (classical, see, e.g., [27]) moment identity and do not depend on $\gamma$, as stated in the following lemma. Note that we take $\gamma > 0$, since $\text{Fix}(\mathcal{M}_0) = E$.

**Lemma 1.** *Suppose that Assumption (H1) holds, and let $\gamma \in (0, 1]$. For all $\eta \in E$, there is $\eta \in \text{Fix}(\mathcal{M}_\gamma)$ if and only if $\mathbb{E}_{q_\eta}[S] = \mathbb{E}_{\check{\varphi}_\eta^\alpha}[S]$.*

From now on, we simply denote $\text{Fix}(\mathcal{M})$ the common set of fixed points of $\mathcal{M}_\gamma$, $\gamma \in (0, 1]$. Finally, we state assumptions that ensure the convergence of (6), which happens at a geometric rate.

**Assumption (H1)** (Variational family). $\mathcal{Q}$ is a regular minimal exponential family of probability densities as in (4).

**Assumption (H2)** (Non-divergence). There exists a compact set $\mathsf{K} \subset E$ such that terms in the sequence defined by $\eta_0 \in \mathsf{K}$ and $\eta_{t+1} = \mathcal{M}_{\gamma_t}(\eta_t)$ for $t \ge 1$ all belong to $\mathsf{K}$, with $(\gamma_t)_{t \ge 1} \in (0, 1]^{\mathbb{N}^*}$.

**Assumption (H3)** (Covariance condition). For all $\eta \in \text{Fix}(\mathcal{M}) \cap \mathsf{K}$, there is

$$\rho_\eta := \alpha \varrho\left( \text{Cov}_{q_\eta}(S)^{-1} \text{Cov}_{\check{\varphi}_\eta^\alpha}(S) \right) < 1, \tag{14}$$

where, for any square matrix $H$, we denote by $\varrho(H)$ its spectral radius.

**Assumption (H4)** (Variability condition). There exists $\delta \in (0, 1]$ such that $\gamma_t \ge \delta$ eventually.

**Theorem 1.** *If Assumptions (H1) to (H4) hold, then the sequence $(\eta_t)$ converges to some parameter $\eta_\star \in \mathrm{Fix}(\mathcal{M}) \cap \mathsf{K}$ that is a strict local minimizer of $\mathcal{L}_\alpha$. Moreover, for all $\rho > 1 - \delta(1 - \rho_\star)$, we have $\|\eta_t - \eta_\star\| = \mathcal{O}(\rho^t)$ as $t \to +\infty$.*

In Theorem 1, we set $\rho_\star = \rho_{\eta_\star}$, following the convention specified in Section 2. We will discuss the assumptions hereafter, but note already that (H2) and (H3) imply $\rho_\star < 1$ and thus $1 - \delta(1 - \rho_\star) < 1$ for all $\delta \in (0, 1]$. Hence, Theorem 1 provides a decreasing geometric rate which can at best be arbitrarily close to $\rho_\star$ if $\delta = 1$. A proof of Theorem 1 is provided in Appendix B.1. Let us now discuss the feasibility and reasonableness of the assumptions.

Assumption (H1) is a matter of model design. We refer to [37] for interesting insight on (regular and minimal) exponential families. For instance, [37, Theorem 3.3] guarantees that $F$ is an open convex set, as the interior of a convex set. Along with definition (13), this implies that for all $\eta \in E$, the set $\{\gamma \in [0, 1], \eta \in E_\gamma\}$ is either $[0, 1]$ if $\mathcal{R}(\eta) \in F$, or of the form $[0, \overline{\gamma}_\eta)$ for some $\overline{\gamma}_\eta > 0$. In other words, $(E_\gamma)_{\gamma \in [0,1]}$ is a non-increasing collection of subsets of $E$, with $E_0 = E$ and $E_1 = \mathcal{R}^{-1}(F)$.

Assumption (H2) specifies the construction of the sequence $(\eta_t)$ in order to satisfy (6). This in particular requires choosing $\gamma_t \in (0, 1]$ such that $\eta_t \in E_{\gamma_t}$, that is, by taking $\gamma_t$ small enough. The compactness of $\mathsf{K}$ is useful to ensure the existence of subsequential limits of the sequence $(\eta_t)$. In practice, given a compact set $\mathsf{K}$, one can choose an initial $\eta_0 \in \mathrm{int}(\mathsf{K})$ and take each $\gamma_t$ small enough to stay in this interior. Alternatively, one can choose any initial $\eta_0 \in E$, and a sequence $(\gamma_t)$ only to satisfy the minimal condition that $\eta_t \in E_{\gamma_t}$ at each step. The second strategy is easier to implement as it does not require choosing a compact set *a priori*, but there is no guarantee that $(\eta_t)$ will remain bounded. The first strategy is safer from this point of view, but a poor choice of $\mathsf{K}$ may lead to vanishing gains $(\gamma_t)$, making it incompatible with (H4).

Assumption (H3) is a condition on the parameters in $\mathrm{Fix}(\mathcal{M})$, so it ultimately depends on $p$ and $\mathcal{Q}$. The proof of Theorem 1 relies heavily on the equivalence provided in Lemma 1, and yet the fixed point of the mappings $(\mathcal{M}_\gamma)$ possess quite a few other remarkable properties. The following proposition elucidates the connections between (H3) and the behavior of the mapping $\mathcal{M}_\gamma$ around the minimizers of $\mathcal{L}_\alpha$.

**Proposition 1.** *Suppose that (H1) holds and let $\eta_\star \in \mathrm{Fix}(\mathcal{M}_\gamma)$. Define the norm $\|\cdot\|_\star$ for $x \in \mathbb{R}^d$ by $\|x\|_\star = \left\|\mathrm{Cov}_{\eta_\star}(S)^{1/2}\, x\right\|$ and let $\rho_\star = \rho_{\eta_\star}$ be defined as in (14). Then, for all $\gamma \in (0, 1]$ and any norm $\|\cdot\|_\bullet$ on $\mathbb{R}^d$, we have*

$$\limsup_{\substack{\eta, \eta' \to \eta_\star \\ \eta \neq \eta'}} \frac{\|\mathcal{M}_\gamma(\eta) - \mathcal{M}_\gamma(\eta')\|_\star}{\|\eta - \eta'\|_\star} = 1 - \gamma(1 - \rho_\star) \leq \limsup_{\substack{\eta, \eta' \to \eta_\star \\ \eta \neq \eta'}} \frac{\|\mathcal{M}_\gamma(\eta) - \mathcal{M}_\gamma(\eta')\|_\bullet}{\|\eta - \eta'\|_\bullet}. \quad (15)$$

*Moreover, the following two assertions are equivalent.*

   *(i) We have $\rho_\star < 1$.*

   *(ii) There exists a norm on $\mathbb{R}^d$ for which the mapping $\mathcal{M}_\gamma$ is a contraction in a neighborhood of $\eta_\star$ for all $\gamma \in (0, 1]$.*

*They also imply the following one.*

   *(iii) $\eta_\star$ is a strict local minimizer of $\mathcal{L}_\alpha$.*

In light of this result, we better understand why choosing $\delta = 1$ in (H4) will yield the fastest asymptotic convergence rate of $\rho_\star$. The only purpose of the sequence $(\gamma_t)$ in the ideal setting is to confine the iterates in a bounded region of the parameter space. Showing that we can choose $(\gamma_t)$ as in (H4) is beyond the scope of this paper, but the intuitive idea is that when close enough to a minimizer of the alpha-divergence, the iterates are always contained in some compact set regardless of the choice of $\gamma_t$ as $t$ goes to infinity.

The next section is about transposing the exact algorithm to an empirical setting, which will naturally lead to formulating Robbins-Monro algorithms. The usual gains conditions $\sum \gamma_t = +\infty$ and $\sum \gamma_t^2 < +\infty$ imply $\gamma_t \longrightarrow 0$, so we will find ourselves in a situation where $\delta = 0$. Combining this observation with insight from [13, Theorems 2 and 3] could explain why we do not get a geometric convergence rate for the empirical version of the algorithm.

# 4 Unbiased alpha-divergence minimization

## 4.1 Justification of the unbiased algorithm

Perhaps the easiest way to get rid of the bias in (9) is to multiply the gradient estimator by $\hat{\ell}_\alpha^*(\mu_t)$, leading to the recursion (11), which we recall is given by

$$\mu_{t+1} = \mu_t + \gamma_t \left[ \hat{\mathcal{E}}(\mu_t) - \mu_t \hat{\ell}_\alpha^*(\mu_t) \right].$$

Taking $h(\mu) = \mathcal{E}(\mu) - \mu \ell_\alpha^*(\mu)$ and $r_{t+1} = \hat{\mathcal{E}}(\mu_t) - \mu_t \hat{\ell}_\alpha^*(\mu_t) - h(\mu_t)$, this can be written as a Robbins-Monro procedure (10). Clearly, denoting by $(\mathcal{F}_t)$ the filtration generated by the random variables simulated up to the computation of $\mu_t$, we have $\mathbb{E}[r_{t+1}|\mathcal{F}_t] = 0$. Note that the zeros of $h$ are characterized by the same moment identity as the one characterizing the fixed point of $\mathcal{M}_\gamma$ in Lemma 1.

Another beneficial property of (11) is that if $\gamma_t$ is small enough to have $\gamma_t/\ell_\alpha^*(\mu_t) < 1$, the *exact* version of this update carries the same monotonicity property as (6).

Interestingly, we can consider (11) as a gradient descent scheme. Indeed, letting $\mathcal{L}_\alpha^*$ be such that $\mathcal{L}_\alpha = \mathcal{L}_\alpha^* \circ (\partial A)$, the gradient of $\mathcal{L}_\alpha^*$ computed with respect to $\eta = (\partial A)^{-1}(\mu)$ is

$$\partial_\eta \mathcal{L}_\alpha^*(\mu) = \frac{1}{\alpha - 1} \int_Y \left[ S(y) - \partial A(\eta) \right] q_\eta(y)^\alpha p(y)^{1-\alpha} \nu(\mathrm{d}y) = \frac{1}{\alpha - 1} \left[ \mathcal{E}(\mu) - \mu \ell_\alpha^*(\mu) \right]. \quad (16)$$

For exponential families that satisfy (H1), the chain rule leads to $\partial_\eta \mathcal{L}_\alpha^*(\mu) = \mathbf{F}(\mu) \partial_\mu \mathcal{L}_\alpha^*(\mu)$, where $\mathbf{F}(\mu)$ is the Fisher information matrix of the model evaluated at $\mu$, i.e., the Hessian matrix of the log-partition $A$ taken at $\eta$. Setting $\gamma_t \leftarrow \gamma_t(\alpha - 1)$, we may write the expectation version of (11) as

$$\mu_{t+1} = \mu_t + \gamma_t \mathbf{F}(\mu_t) \partial_\mu \mathcal{L}_\alpha^*(\mu_t).$$

Under (H1), the matrix $\mathbf{F}(\mu_t)$ is positive definite, hence the unbiased algorithm appears as a gradient descent procedure on $\mathcal{L}_\alpha^*$. More specifically, it is a steepest descent algorithm for the dissimilarity measure $D(\mu, \mu') = \|\mu - \mu'\|_{\mathbf{F}^{-1}(\mu)} = (\mu - \mu')^\top \mathbf{F}^{-1}(\mu)(\mu - \mu')$. In other words, the update writes

$$\mu_{t+1} = \arg\min_{\boldsymbol{m} \in \mathbb{R}^d} \left\{ (\boldsymbol{m} - \mu_t)^\top \partial_\mu \mathcal{L}_\alpha^*(\mu_t) + \frac{1}{2\gamma_t} D(\mu_t, \boldsymbol{m}) \right\}.$$

Lastly, note that we could obtain similar guarantees for natural gradient procedures [1, 2, 17], that is, $\eta_{t+1} = \eta_t + \gamma_t \partial_\mu \mathcal{L}_\alpha(\eta_t)$, or with plain gradient descent, i.e., $\mu_{t+1} = \mu_t + \gamma_t \partial_\mu \mathcal{L}_\alpha^*(\mu_t)$ or $\eta_{t+1} = \eta_t + \gamma_t \partial_\eta \mathcal{L}_\alpha(\eta_t)$. However, these algorithms either require evaluating the inverse of the Fisher information matrix – which can be computationally expensive, with an $\mathcal{O}(d^3)$ cost –, or they are unstable in practice due to unfavorable loss landscapes (see the toy examples in Section 6). On top of that, we lose the monotonicity property for the exact versions of these procedures.

These four arguments justify the choice of algorithm (11), which is ultimately quite close to (6). However, obtaining guarantees for this new algorithm is far easier, as covered in the next section.

## 4.2 Convergence results

We now state two theorems regarding the convergence of algorithm (11). Let us start by giving some definitions. A matrix $H \in \mathbb{R}^{d \times d}$ is said to be stable if, for all $\lambda \in \mathrm{Sp}(H)$, we have $\Re(\lambda) < 0$, i.e., the eigenvalues of $H$ have negative real parts. Then, we denote $L(H) = -\max\{\Re(\lambda), \lambda \in \mathrm{Sp}(H)\}$. We also still denote $F = \partial A(E)$, and introduce the critical set $\mathcal{S} = \{\mu \in F, \mathcal{E}(\mu) = \mu \ell_\alpha^*(\mu)\}$. The set $\mathcal{S}$ is qualified as critical, since $\mu \in \mathcal{S}$ if and only if $\partial_\mu \mathcal{L}_\alpha^*(\mu) = 0$. Remember that (11) can be written as a special case of (10) by taking $h(\mu) = \mathcal{E}(\mu) - \mu \ell_\alpha^*(\mu)$ and $r_{t+1} = \hat{\mathcal{E}}(\mu_t) - \mu_t \hat{\ell}_\alpha^*(\mu_t) - h(\mu_t)$. Lastly, given some sequence $(\mu_t) \in F^\mathbb{N}$, a point $\mu_\star \in F$, and a subset $\mathsf{M} \subset F$, we define the events

$$\Gamma(\mu_\star) = \{\omega \in \Omega, \mu_t(\omega) \longrightarrow \mu_\star(\omega)\} \quad \text{and} \quad \Xi(\mathsf{M}) = \{\omega \in \Omega, \exists t_0 \geq 0, \forall t \geq t_0, \mu_t(\omega) \in \mathsf{M}\},$$

and we denote $H_\star \in \mathbb{R}^{d \times d}$ the Jacobian matrix of $h$ at $\mu_\star$.

**Assumption (C0)** (Gains). The positive sequence $(\gamma_t)$ is such that $\sum \gamma_t = +\infty$ and $\sum \gamma_t^2 < +\infty$.

**Assumption (C1)** (Critical set). The elements in the set $\mathcal{S}$ are isolated, i.e., for all $\mu \in \mathcal{S}$, there exists an open neighborhood $\mathcal{V}$ of $\mu$ such that $\mathcal{V} \cap \mathcal{S} = \{\mu\}$.

**Assumption (C2)** (Moment condition). The set M is compact and, with $\eta = (\partial A)^{-1}(\mu)$, there is

$$\sup_{\mu \in \mathsf{M}} \mathbb{E}_{q_\eta} \left[ \|S - \mu\|^2 \left( \frac{p}{q_\eta} \right)^{2(1-\alpha)} \right] < +\infty.$$

The following theorem guarantees the almost sure convergence of the sequence (11) to some $\mu_\star$, given the existence of a compact subset of $F$ to which the terms in $(\mu_t)$ belong eventually.

**Theorem 2** (Almost sure convergence). *Let $\mathcal{Q}$ be an exponential family of densities as in (H1), $(\gamma_t)$ a positive sequence that satisfies (C0), and M a (compact) subset of $F$. We construct a sequence $(\mu_t)$ using recursion (11). If conditions (C1) and (C2) hold, with $\mathcal{S} \cap \mathsf{M} \neq \emptyset$, then*

$$\mathbb{P} \left( \left. \bigcup_{\mu_\star \in \mathcal{S} \cap \mathsf{M}} \Gamma(\mu_\star) \; \right| \; \Xi(\mathsf{M}) \right) = 1.$$

For the next result, we need to strengthen Assumptions (C0) and (C2) slightly. Recall that $H_*$ denotes the Jacobian matrix of $h$ at $\mu_*$.

**Assumption (C0')** (Gains). There exists $\delta \in (\frac{1}{2}, 1]$ such that for all $t \geq 1$, we have $\gamma_t = \gamma_0 t^{-\delta}$, with $\gamma_0 > 0$ if $\delta \in (\frac{1}{2}, 1)$, or $\gamma_0 > \beta/(2L(H_\star))$ for some arbitrary $\beta \in (0, 1]$ if $\delta = 1$ and $H_\star$ is stable.

**Assumption (C2')** (Additional moment condition). We can find $R > 0$ and $b > 2$ such that for all $\mu_\star \in \mathcal{S}$,

$$\sup_{\substack{\mu \in F \\ \|\mu - \mu_\star\| \leq R}} \mathbb{E}_{q_\eta} \left[ \|S - \mu\|^b \left( \frac{p}{q_\eta} \right)^{b(1-\alpha)} \right] < +\infty.$$

The result of Theorem 3 holds almost surely on the event $\Gamma(\mu_\star)$, thus complementing Theorem 2 by giving an almost sure convergence rate.

**Theorem 3** (Law of the iterated logarithm). *Construct a sequence $(\mu_t)$ as in (11) using the same number of samples $K$ at each iteration, and let $\mu_\star \in \mathcal{S}$. Assume that (C0'), (C2') and (H3) hold, with $\delta > 2/b$. Then, letting $S_t = \sum_{i=1}^{t} \gamma_i$, there exists a real constant $\Lambda$ such that*

$$\limsup \sqrt{\frac{t}{\ln(S_t)}} \|\mu_t - \mu_\star\| \leq \Lambda \quad \textit{almost surely on } \Gamma(\mu_\star).$$

Let us now discuss these two results. Theorem 2 guarantees the convergence of (11) to a critical point of $\mathcal{L}_\alpha^*$, i.e., a minimizer of the alpha-divergence when (H3) is also satisfied. However, it gives no information regarding the minimizer to which the algorithm will converge. It is a typical convergence result for Robbins-Monro algorithms, with the added difficulty that there are many possible points of convergence. The proof given in Appendix B.2 consists in showing that the distance between $\mu_t$ and $\mathcal{S} \cap \mathsf{M}$ almost surely tends to 0. Then, we use (C0) and (C1) to show that the iterates cannot jump from one critical point to another.

The result of Theorem 2 is stated conditionally to the event $\Xi(\mathsf{M})$. While we immediately notice if the iterates get out of M in practice, finding *a priori* a compact set that has a non-empty intersection with $\mathcal{S}$ and that almost surely contains the terms in $(\mu_t)$ is virtually impossible, since it would require knowing $h$. However, under mild assumptions on $\mathcal{L}_\alpha^*$, there exists an elegant and theoretically sound way around this problem. Chen's algorithm [4] consists in taking an increasing sequence of compact sets $(\mathsf{M}_s)_{s \geq 0}$ that eventually covers the entire parameter space. If the update flies out of the current compact, say $\mathsf{M}_s$, we reset the sequence of iterates by randomly choosing an initial parameter in $\mathsf{M}_0$ and start over, but with a larger compact subset of $E$, namely $\mathsf{M}_{s+1}$. To simplify the exposition and provide easy-to-implement algorithms, we will only consider that we are given a compact set M that eventually contains the iterates. However, with some minor tweaks to our assumptions, it could be possible to obtain similar guarantees with Chen's algorithm (e.g., by adapting results from [11]).

Finally, let us briefly discuss assumptions (C1) and (C2). The former is verified if the Hessian matrix of $\mathcal{L}_\alpha$ is definite positive at all $\eta \in (\partial A)^{-1}(\mathcal{S})$ (see Proposition 1 and its proof in Appendix B.1). In

particular, (C1) is implied by (H3). As for (C2), notice that for $\alpha \in [1/2, 1)$, by Jensen's inequality, it is satisfied as soon as $\int \|S\|^{1/(1-\alpha)} p \, d\nu < \infty$. Extending it to $\alpha \in (0, 1/2)$ supposes specific behaviors on the relative tails of $p$ and $q_\eta$.

## 5 Related algorithms in the context of Variational Auto-Encoders

In this section, we explain how to transpose the algorithms presented in Section 4.1 to the training of Variational Auto-Encoders (VAEs) [21]. We start by showing that the exact versions of the biased and unbiased algorithms correspond to gradient ascent procedures on two different variational bounds. Let us first address the case of the biased algorithm. Recall that it writes

$$\mu_{t+1} = \mu_t + \gamma_t \left[ \frac{\mathcal{E}(\mu_t)}{\ell_\alpha^*(\mu_t)} - \mu_t \right].$$

For $\eta \in E$ and $\mu = \partial A(\eta)$, we define the Variational Rényi (VR) bound [25] by

$$\mathcal{L}_\alpha^{\mathrm{R}}(\mu) = \frac{1}{1-\alpha} \log \mathbb{E}_{y \sim q_\eta} \left[ \left( \frac{p(y)}{q_\eta(y)} \right)^{1-\alpha} \right] = \frac{1}{1-\alpha} \log \ell_\alpha^*(\mu). \tag{17}$$

Noticing that $\partial_\eta \ell_\alpha^*(\mu) = \alpha \left( \mathcal{E}(\mu) - \mu \ell_\alpha^*(\mu) \right)$, we can write an iteration of the biased algorithm as

$$\mu_{t+1} = \mu_t + \frac{\gamma_t \cdot \alpha}{1-\alpha} \partial_\eta \mathcal{L}_\alpha^{\mathrm{R}}(\mu_t) = \mu_t + \frac{\gamma_t \cdot \alpha}{1-\alpha} \mathbf{F}(\mu_t) \partial_\mu \mathcal{L}_\alpha^{\mathrm{R}}(\mu_t).$$

Under (H1), the Fisher Information Matrix $\mathbf{F}(\mu_t)$ is positive definite, hence the biased algorithm is a gradient ascent procedure on the VR bound.

The unbiased algorithm, which writes $\mu_{t+1} = \mu_t + \gamma_t \left[ \mathcal{E}(\mu_t) - \mu \ell_\alpha^*(\mu_t) \right]$, similarly amounts to performing gradient ascent on the variational bound $\mathcal{L}_\alpha^{\mathrm{G}}$ defined by

$$\mathcal{L}_\alpha^{\mathrm{G}}(\mu) = \frac{1}{1-\alpha} \mathbb{E}_{y \sim q_\eta} \left[ \left( \frac{p(y)}{q_\eta(y)} \right)^{1-\alpha} \right] = \frac{1}{1-\alpha} \ell_\alpha^*(\mu). \tag{18}$$

In the context of VAEs, we learn both a probabilistic encoder $y \mapsto \tilde{q}_{f(x;\eta)}(y)$ and a probabilistic decoder $x \mapsto \tilde{p}_{g(y;\theta)}(x)$, where $\tilde{q}_\eta$ and $\tilde{p}_\theta$ are densities from families parameterized respectively by $\eta$ and $\theta$, while $x \mapsto f(x;\eta)$ and $y \mapsto g(y;\theta)$ are neural networks. For simplicity and to align with the usual notation for VAEs, we will denote $\tilde{q}_{f(x;\eta)}(\cdot) = q_\eta(\cdot \,|\, x)$ and $\tilde{p}_{g(y;\theta)}(\cdot) = p_\theta(\cdot \,|\, y)$. We will also use the shorthand $\phi = (\eta, \theta)$.

Since the biased and unbiased algorithms studied in the previous sections minimize the alpha-divergence by maximizing the variational bounds $\mathcal{L}_\alpha^{\mathrm{R}}$ and $\mathcal{L}_\alpha^{\mathrm{G}}$, we propose to train VAEs to maximize those same bounds. In this new setting, they write

$$\mathcal{L}_\alpha^{\mathrm{R}}(\eta, \theta, x) = \frac{1}{1-\alpha} \log \mathbb{E}_{y \sim q_\eta(\cdot \,|\, x)} \left[ \left( \frac{p_\theta(x, y)}{q_\eta(y \,|\, x)} \right)^{1-\alpha} \right],$$

$$\mathcal{L}_\alpha^{\mathrm{G}}(\eta, \theta, x) = \frac{1}{1-\alpha} \mathbb{E}_{y \sim q_\eta(\cdot \,|\, x)} \left[ \left( \frac{p_\theta(x, y)}{q_\eta(y \,|\, x)} \right)^{1-\alpha} \right].$$

To update both the encoder and decoder simultaneously, we differentiate them with respect to $\phi$ using the reparameterization trick [21]. If $z \sim r(\cdot)$ and there exists a mapping $v$ such that $v(z; \eta, x)$ has the same distribution as $y$ when $y \sim q_\eta(\cdot \,|\, x)$, then we have

$$\partial_\phi \mathcal{L}_\alpha^{\mathrm{R}}(\eta, \theta, x) = \mathbb{E}_{z \sim r(\cdot)} \left[ \overline{w}_\alpha(z, \eta, x) \partial_\phi \left( \log \frac{p_\theta(x, v(z; \eta, x))}{q_\eta(v(z; \eta, x) \,|\, x)} \right) \right], \tag{19}$$

$$\partial_\phi \mathcal{L}_\alpha^{\mathrm{G}}(\eta, \theta, x) = \mathbb{E}_{z \sim r(\cdot)} \left[ w_\alpha(z, \eta, x) \partial_\phi \left( \log \frac{p_\theta(x, v(z; \eta, x))}{q_\eta(v(z; \eta, x) \,|\, x)} \right) \right]. \tag{20}$$

where $w_\alpha(z, \eta, x) = \left( \dfrac{p_\theta(x, v(z; \eta, x))}{q_\eta(v(z; \eta, x) \,|\, x)} \right)^{1-\alpha}$ and $\overline{w}_\alpha(z, \eta, x) = \dfrac{w_\alpha(z, \eta, x)}{\int_Y w_\alpha(z', \eta, x) r(z') \nu(\mathrm{d}z')}$.

To train VAEs, we simply plug batch estimates of these gradients into an optimizer like Adam. Notably, $\partial_\phi \mathcal{L}_\alpha^{\mathrm{G}}$ can be estimated unbiasedly, while estimators of $\partial_\phi \mathcal{L}_\alpha^{\mathrm{R}}$ are subject to bias. We will study the practical implications of this fact in Section 6.

## 6  Experiments

**Toy Gaussian.**  For our toy experiments in the empirical case, we keep a Gaussian variational family. We try to approximate various targets, including a Gaussian mixture and a Cauchy distribution. The latter does not verify assumption (C2), as its tails are too heavy. Even then, the studied algorithms seem to converge without any issue. Let us now focus on the case of the Gaussian Mixture. On the figures, MAX, UNB, SGE, SGM and NAT respectively stand for maximization approach (9), unbiased algorithm (11), SGD on $\eta$, SGD on $\mu$, and natural gradient descent (as discussed at the end of Section 4.1). Since the parameter space has only two dimensions, we also plot the loss landscape and the trajectories followed by the different algorithms (e.g. Gaussian mixture target in Figure 1). We observe that MAX and UNB always seem to have a straightforward path toward a local minimizer of the alpha-divergence, while the other procedures are heavily influenced by the loss landscapes in their respective parameter spaces. Unfavorable landscapes can lead to erratic trajectories and slow convergence, or even to convergence failure (see Appendix D). The MAX and UNB algorithms exhibit better stability and reliability on these toy examples.

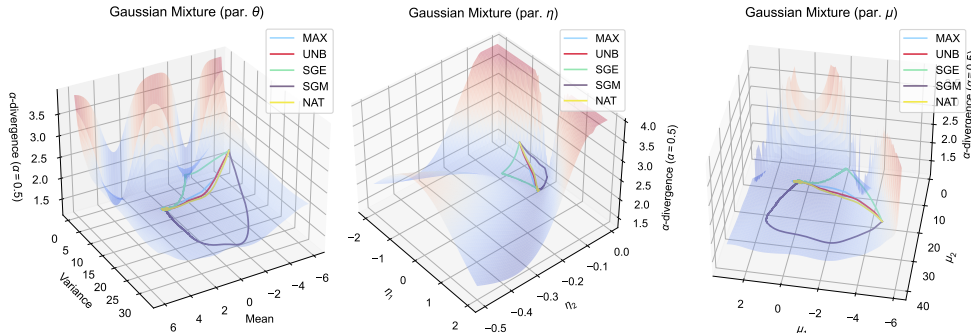

Figure 1: Trajectories to approximate a Gaussian mixture with a Gaussian family, represented under different parameterizations.

On Figure 1, we approximate a Gaussian mixture with a single Gaussian. With $\alpha = 0.5$, the landscape of the alpha-divergence around the starting point has three local minima. The global optimum is situated on the left in the first plot. It is not attained by any of the algorithms, as they all converge to the local minimum in the center. The stochastic gradient with respect to $\eta$ (SGE, in green) almost gets attracted to the local minimum on the right in the first plot, which is the worst of all three. More precisely, Figure 2 shows boxplots of the alpha-divergence after a certain number of iterations for all five algorithms. The number of Monte-Carlo samples is set to 10, and we run each algorithm 100 times. We see that the MAX approach is the quickest to converge, however it lands to slightly suboptimal parameters and gets surpassed by UNB, SGE, and even SGM after a sufficient number of iterations. This could be due to the bias induced by the normalization, as discussed in Section 4.

**Variational Auto-Encoders.**  We evaluate the two approaches described in Section 5 on the image datasets CIFAR10 ($50\,000$ images of size $32 \times 32$) and CelebA ($192\,599$ randomly chosen training images cropped to $128 \times 128$), comparing their performance for varying values of $\alpha$. VAEs have achieved notable success in separating style and content on various types of data, like images and time series. However, they may suffer posterior collapse on challenging datasets as a result of learning an entangled latent representation. Several authors have investigated ways to disentangle the latent space (see, e.g., [29]), and alpha-divergence minimization could be helpful to learn suitable latent representations. In this series of experiments, we train VAEs with a Gaussian prior to generate new images, the encoders consisting of cascades of four convolutional layers, with batch normalization

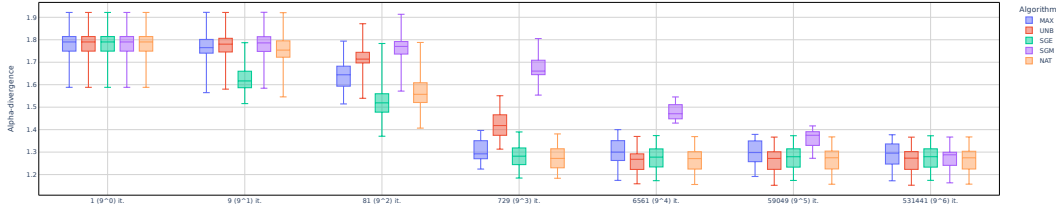

Figure 2: Boxplot of the alpha-divergence against the number of iterations for the five algorithms.

and leaky ReLU activations. Similarly, the decoders are composed of four transposed convolutional layers. We set the latent dimensions to $64$ on CIFAR10 and $128$ on CelebA.

We compute the gradient estimators (19) and (20). These approaches are respectively referred to as VR and UB in Table 1. We choose the number of samples to be $K = 5$. The weights are optimized with Adam [20], using learning rates of $8\mathrm{e}-4$ on CIFAR10 and $2\mathrm{e}-4$ on CelebA, and $(\beta_1, \beta_2) = (0.9, 0.999)$ in both cases. We set the batch size to $256$ and train for $100$ epochs on CIFAR10 and $30$ epochs on CelebA, for a total of roughly $20\,000$ iterations on both datasets. Training takes $30$ minutes per model on CIFAR10 and a few hours on CelebA, using a single V100 GPU. To evaluate the models, we use the Fréchet Inception Distance (FID) metric [16]. Both test sets include $10\,000$ examples.

Table 1: Fréchet Inception Distance (FID) on CIFAR-10 and CelebA. Lower is better.

| Dataset | $\alpha = 0.01$ | | $\alpha = 0.3$ | | $\alpha = 0.5$ | | $\alpha = 0.7$ | | $\alpha = 0.99$ | |
|---|---|---|---|---|---|---|---|---|---|---|
| | VR | UB | VR | UB | VR | UB | VR | UB | VR | UB |
| CIFAR-10 | 102.0 | 85.8 | 102.8 | 95.4 | 87.0 | 90.9 | 87.4 | 108.3 | **81.2** | 110.3 |
| CelebA | 241.4 | 237.6 | 240.0 | 237.7 | 257.7 | 237.6 | 234.3 | **231.1** | 239.7 | 232.5 |

| | $\alpha = 0$ (IWAE) | $\alpha = 1$ (VAE) |
|---|---|---|
| CIFAR-10 | 122.2 | 121.1 |
| CelebA | 243.8 | 235.3 |

It seems that the choice of $\alpha$ can indeed make a difference in the quality of the resulting model, in terms of FID to the test set. The best values of $\alpha$ differ between the datasets, and are also inconsistent between the two approaches. For instance, on CIFAR10, the better results for the VR approach are obtained when $\alpha$ gets closer to $1$, while the opposite is observed for the UB approach. Still, for both approaches and on both datasets, the IWAE and VAE baselines can be outperformed with a proper choice of $\alpha$. Though it is not visible in the FID score, the losses obtained with the UB approach are significantly higher than those yielded by the VR approach, and we find that normalizing the importance weights yields a much stabler training process than using unbiased gradient estimators.

## 7 Conclusion

Our paper presents various asymptotic results for both exact and empirical alpha-divergence minimization algorithms for Variational Inference. We find that the studied algorithms converge to a minimizer of the alpha-divergence. The rate of convergence is asymptotically geometric for the exact procedure, and we prove a law of the iterated logarithm in the empirical setting. However, we only address the case where the variational family is an exponential model. While this covers many use cases, the general case should be investigated in future research.

# References

[1] S.-i. Amari. *Differential-Geometrical Methods in Statistics*, volume 28 of *Lecture Notes in Statistics*. Springer, 1985.

[2] S.-i. Amari. Natural Gradient Works Efficiently in Learning. *Neural Computation*, 10(2):251–276, 1998.

[3] D. M. Blei, A. Kucukelbir, and J. D. McAuliffe. Variational Inference: A Review for Statisticians. *Journal of the American Statistical Association*, 112(518), 2017.

[4] H.-F. Chen, L. Guo, and A.-J. Gao. Convergence and robustness of the Robbins-Monro algorithm truncated at randomly varying bounds. *Stochastic Processes and their Applications*, 27:217–231, 1987.

[5] L. Chen, L. Wang, Z. Han, J. Zhao, and W. Wang. Variational inference based kernel dynamic bayesian networks for construction of prediction intervals for industrial time series with incomplete input. *IEEE/CAA Journal of Automatica Sinica*, 7(5):1437–1445, 2019.

[6] A. Cichocki and S. Amari. Families of Alpha- Beta- and Gamma- Divergences: Flexible and Robust Measures of Similarities. *Entropy*, 12:1532–1568, 2010.

[7] K. Daudel, J. Benton, Y. Shi, and A. Doucet. Alpha-divergence Variational Inference Meets Importance Weighted Auto-Encoders: Methodology and Asymptotics. *Journal of Machine Learning Research*, 24(243):1–83, 2023.

[8] K. Daudel, R. Douc, and F. Portier. Infinite-dimensional gradient-based descent for alpha-divergence minimisation. *The Annals of Statistics*, 49(4):2250 – 2270, 2021.

[9] K. Daudel, R. Douc, and F. Roueff. Monotonic Alpha-Divergence Minimisation for Variational Inference. *Journal of Machine Learning Research*, 24(62):1–76, 2023.

[10] K. Daudel et al. Mixture weights optimisation for alpha-divergence variational inference. *Advances in Neural Information Processing Systems*, 34:4397–4408, 2021.

[11] B. Delyon. Stochastic approximation with decreasing gain: Convergence and asymptotic theory. *Unpublished lecture notes, Université de Rennes*, 26:39, 2000.

[12] J. Domke, R. Gower, and G. Garrigos. Provable convergence guarantees for black-box variational inference. *Advances in Neural Information Processing Systems*, 36, 2024.

[13] R. Douc and S. L. Corff. Asymptotic convergence of iterative optimization algorithms. *arXiv preprint arXiv:2302.12544*, 2023.

[14] T. Geffner and J. Domke. On the difficulty of unbiased alpha divergence minimization. In M. Meila and T. Zhang, editors, *Proceedings of the 38th International Conference on Machine Learning*, volume 139 of *Proceedings of Machine Learning Research*, pages 3650–3659. PMLR, 18–24 Jul 2021.

[15] J. Hernandez-Lobato, Y. Li, M. Rowland, T. Bui, D. Hernández-Lobato, and R. Turner. Black-box alpha divergence minimization. In *International conference on machine learning*, pages 1511–1520. PMLR, 2016.

[16] M. Heusel, H. Ramsauer, T. Unterthiner, B. Nessler, and S. Hochreiter. Gans trained by a two time-scale update rule converge to a local nash equilibrium. *Advances in Neural Information Processing Systems*, 30, 2017.

[17] M. D. Hoffman, D. M. Blei, C. Wang, and J. Paisley. Stochastic Variational Inference. *Journal of Machine Learning Research*, 14(40):1303–1347, 2013.

[18] D. R. Hunter and K. Lange. A Tutorial on MM Algorithms. *The American Statistician*, 58(1):30–37, 2004.

[19] M. I. Jordan, Z. Ghahramani, T. S. Jaakkola, and L. K. Saul. An introduction to variational methods for graphical models. *Machine learning*, 37:183–233, 1999.

[20] D. P. Kingma and J. Ba. Adam: a method for stochastic optimization (2014). *arXiv preprint arXiv:1412.6980*, 15, 2017.

[21] D. P. Kingma and M. Welling. Auto-encoding Variational Bayes. *arXiv preprint arXiv:1312.6114*, 2013.

[22] D. P. Kingma and M. Welling. An Introduction to Variational Autoencoders. *Foundations and Trends in Machine Learning*, 12(4):307–392, 2019.

[23] K. Lange, D. R. Hunter, and I. Yang. Optimization Transfer Using Surrogate Objective Functions. *Journal of Computational and Graphical Statistics*, 9(1), 2000.

[24] K. Lange, J. Won, A. Landeros, and H. Zhou. Nonconvex Optimization via MM Algorithms: Convergence Theory. *Wiley StatsRef: Statistics Reference Online*, page 1–22, 2021.

[25] Y. Li and R. E. Turner. Rényi Divergence Variational Inference. In *Advances in Neural Information Processing Systems*, volume 29, 2016.

[26] X. Liu and S. Sun. Alpha-divergence minimization with mixed variational posterior for bayesian neural networks and its robustness against adversarial examples. *Neurocomputing*, 423:427–434, 2021.

[27] T. Minka et al. Divergence measures and message passing. Technical report, Microsoft Research, 2005.

[28] C. Naesseth, F. Lindsten, and D. Blei. Markovian score climbing: Variational inference with KL(p‖ q). *Advances in Neural Information Processing Systems*, 33:15499–15510, 2020.

[29] K. Oublal, S. Ladjal, D. Benhaiem, E. L. BORGNE, and F. Roueff. Disentangling time series representations via contrastive independence-of-support on l-variational inference. In *The Twelfth International Conference on Learning Representations*, 2024.

[30] M. Pelletier. On the almost sure asymptotic behaviour of stochastic algorithms. *Stochastic processes and their applications*, 78(2):217–244, 1998.

[31] J.-B. Regli and R. Silva. Alpha-beta divergence for variational inference. *arXiv preprint arXiv:1805.01045*, 2018.

[32] A. Rényi. On measures of entropy and information. In *Proceedings of the fourth Berkeley symposium on mathematical statistics and probability, volume 1: contributions to the theory of statistics*, volume 4, pages 547–562. University of California Press, 1961.

[33] H. Robbins and S. Monro. A Stochastic Approximation Method. *The Annals of Mathematical Statistics*, 22, 1951.

[34] C. P. Robert and G. Casella. *Monte Carlo Statistical Methods*. Springer Texts in Statistics. Springer, 2004.

[35] S. Rodriguez-Santana and D. Hernández-Lobato. Adversarial $\alpha$-divergence minimization for Bayesian approximate inference. *Neurocomputing*, 471:260–274, 2022.

[36] T. Van Erven and P. Harremos. Rényi divergence and Kullback-Leibler divergence. *IEEE Transactions on Information Theory*, 60(7):3797–3820, 2014.

[37] M. J. Wainwright and M. I. Jordan. Graphical models, exponential families, and variational inference. *Foundations and Trends® in Machine Learning*, 1(1–2):1–305, 2008.

[38] Y. Yang, D. Pati, and A. Bhattacharya. $\alpha$-variational inference with statistical guarantees. *The Annals of Statistics*, 48(2):886 – 905, 2020.

[39] L. Zhang, D. M. Blei, and C. A. Naesseth. Transport score climbing: Variational inference using forward KL and adaptive neural transport. *arXiv preprint arXiv:2202.01841*, 2022.

# A  Table of notation

| Symbol | Description | Expression |
|---|---|---|
| $\circ$ | Function composition | |
| $p$ | Target | |
| $q_\eta$ | Variational density | $q_\eta(y) = \kappa(y)\exp\left[\langle S(y),\eta\rangle - A(\eta)\right]$ |
| $E$ | Set of variational parameters | $E = \{\eta \in \mathbb{R}^d, |A(\eta)| < +\infty\}$ |
| $\mathcal{Q}$ | Variational family | $\mathcal{Q} = \{q_\eta, \eta \in E\}$ |
| $\nu(f)$ | Integral of $f$ w.r.t. $\nu$ | $\nu(f) = \int f\,\mathrm{d}\nu$ |
| $\mathbb{E}_f[g]$ | Expectation of $g$ under the density $f$ | $\mathbb{E}_f[g] = \int gf\,\mathrm{d}\nu$ |
| $\mathrm{Cov}_f[g]$ | Covariance of $g$ under the density $f$ | $\mathrm{Cov}_f[g] = \int g(g - \mathbb{E}_f[g])f\,\mathrm{d}\nu$ |
| $\partial_x f$ | Gradient of $f$ w.r.t. $x$ | |
| $\partial A$ | Gradient of $A$ w.r.t. $\eta$ | $\partial A(\eta) = \mathbb{E}_{q_\eta}[S]$ |
| $\varphi_\eta^\alpha$ | Geometric average of $p$ and $q_\eta$ | $\varphi_\eta^\alpha = q_\eta^\alpha p^{1-\alpha}$ |
| $\ell_\alpha$ | Normalization constant | $\ell_\alpha(\eta) = \int_{\mathsf{Y}} q_\eta(y)p(y)^{1-\alpha}\nu(\mathrm{d}y)$ |
| $\ell_\alpha^*$ | Normalization constant | $\ell_\alpha^* = \ell_\alpha \circ (\partial A)$ |
| $\check{\varphi}_\eta^\alpha$ | Normalized version of $\varphi_\eta^\alpha$ | $\check{\varphi}_\eta^\alpha = \varphi_\eta^\alpha / \ell_\alpha(\eta)$ |
| $\mathcal{L}_\alpha$ | Objective function | $\mathcal{L}_\alpha(\eta) = \dfrac{1}{\alpha(\alpha-1)}\ell_\alpha(\eta)$ |
| $\mathcal{L}_\alpha^*$ | Objective as a function of $\mu$ | $\mathcal{L}_\alpha^* = \mathcal{L}_\alpha \circ (\partial A)$ |
| $\mathcal{R}(\eta)$ | Expectation of $S$ under $\check{\varphi}_\eta^\alpha$ | $\mathcal{R}(\eta) = \mathbb{E}_{\check{\varphi}_\eta^\alpha}[S] = \dfrac{1}{\ell_\alpha(\eta)}\int_{\mathsf{Y}} S(y)\check{\varphi}_\eta^\alpha\nu(\mathrm{d}y)$ |
| $\mu_\bullet$ | Mean of an exponential distribution with natural parameter $\eta_\bullet$ | $\mu_\bullet = \partial A(\eta_\bullet)$ |
| $\mathcal{E}(\mu)$ | Integral of $S\varphi_\eta^\alpha$ w.r.t. $\nu$ | $\mathcal{E}(\mu) = \ell_\alpha(\eta)\cdot\mathcal{R}(\eta)$ |
| $\hat{\mathcal{E}}(\mu; K)$ | Monte-Carlo estimator of $\mathcal{E}(\eta)$ with $K$ i.i.d. samples $y_1,\ldots,y_K \sim q_\eta$ | $\hat{\mathcal{E}}(\mu; K) = \dfrac{1}{K}\sum_{i=1}^{K} S(y_i)\left(\dfrac{p(y_i)}{q_\eta(y_i)}\right)^{1-\alpha}$ |
| $\hat{\ell}_\alpha^*(\mu; K)$ | Monte-Carlo estimator of $\ell_\alpha(\eta)$ with $K$ i.i.d. samples $y_1,\ldots,y_K \sim q_\eta$ | $\hat{\ell}_\alpha^*(\mu; K) = \dfrac{1}{K}\sum_{i=1}^{K}\left(\dfrac{p(y_i)}{q_\eta(y_i)}\right)^{1-\alpha}$ |
| $F$ | Parameter space for the mean parameterization | $F = \partial A(E)$ |
| $\mathcal{M}_\gamma$ | Exact algorithm iterated function | $\mathcal{M}_\gamma : \eta \mapsto (\partial A)^{-1}\left[\gamma\mathcal{R}(\eta) + (1-\gamma)\partial A(\eta)\right]$ |
| $E_\gamma$ | Domain of $\mathcal{M}_\gamma$ | $E_\gamma = \{\eta \in E, \gamma\mathcal{R}(\eta) + (1-\gamma)\partial A(\eta) \in F\}$ |
| $\mathrm{Fix}(\mathcal{M})$ | Set of fixed points of $\mathcal{M}_\gamma$ for any $\gamma$ | $\mathrm{Fix}(\mathcal{M}) = \{\eta \in E, \mathcal{M}_\gamma(\eta) = \eta\}$ |
| $\mathsf{K}$ | Compact subset of $E$ | |
| $\mathsf{M}$ | Compact subset of $F$ | |
| $\rho_\eta$ | Covariance ratio | $\rho_\eta = \alpha\varrho\left(\mathrm{Cov}_{q_\eta}(S)^{-1}\mathrm{Cov}_{\check{\varphi}_\eta^\alpha}(S)\right)$ |
| $\mathbf{F}(\mu)$ | Fisher information matrix of the model $\mathcal{Q}$ at $\eta = (\partial A)^{-1}(\mu)$ | |
| $L(H)$ | Opposite of the largest real part of eigenvalues in the spectrum of the symmetric matrix $H$ | $L(H) = -\max\{\Re(\lambda), \lambda \in \mathrm{Sp}(H)\}$ |
| $\mathcal{S}$ | Critical set of $\mathcal{L}_\alpha^*$ | $\mathcal{S} = \{\mu \in F, \mathcal{E}(\mu) = \mu\mathcal{L}_\alpha^*(\mu)\}$ |
| $\Gamma(\mu_\star)$ | Event on which the random sequence $(\mu_t)$ converges to $\mu_*$ | $\Gamma(\mu_\star) = \{\omega \in \Omega, \mu_t(\omega) \longrightarrow \mu_\star(\omega)\}$ |
| $\Xi(\mathsf{M})$ | Event on which the random sequence $(\mu_t)$ falls into the compact set $\mathcal{M}$ | $\Xi(\mathsf{M}) = \{\omega \in \Omega, \exists t_0 \geq 0, \forall t \geq t_0, \mu_t(\omega) \in \mathsf{M}\}$ |

# B  Proofs

**Preliminaries on useful derivatives.**   We start with some elementary and useful facts on the derivatives of mappings defined through integrals. Recall that, under (H1), the log-partition function $A$ is of class $\mathcal{C}^\infty$ and strictly convex on $E$ with gradient and Hessian given by

$$\partial A(\eta) = \mathbb{E}_{q_\eta}[S] \quad \text{and} \quad \mathbf{H}_A(\eta) = \mathrm{Cov}_{q_\eta}(S)\,, \qquad \eta \in E. \tag{21}$$

The differentiation operations under the integral sign can be justified by the expression of the moment generating function of the sufficient statistic $S$ under the various probabilities at hand. More precisely, for all $\eta$ in the open set $E \subseteq \mathbb{R}^d$ and $\varsigma \in \mathbb{R}^d$ such that $\eta + \varsigma \in E$, we have

$$\mathbb{E}_{q_\eta}\big[\exp(\langle \varsigma, S \rangle)\big] = \exp\big(A(\eta + \varsigma) - A(\eta)\big).$$

This fact is useful to differentiate under the integral mappings such as $\eta \mapsto \mathbb{E}_{q_\eta}\Big[\prod_{j=1}^m S_{k_j}\Big]$ for $m \in \mathbb{N}$ and indices $k_1, \ldots, k_m \in \{1, \ldots, d\}$.

Similar arguments apply with the quantities defined in (2). Since $\int p\,\mathrm{d}\nu$ is a positive constant by definition and $\kappa > 0$ $\nu$-almost everywhere, the Hölder inequality yields, for all $\eta \in E$,

$$0 < \nu\big(\varphi_\eta^\alpha\big) \leq \left(\int p\,\mathrm{d}\nu\right)^{1-\alpha} < +\infty.$$

Thus for all $\eta \in E$ and $\varsigma \in \mathbb{R}^d$ such that $\eta + \varsigma \in E$,

$$\mathbb{E}_{\check\varphi_\eta^\alpha}\big[\exp(\alpha\langle\varsigma, S\rangle)\big] = \frac{\nu\big(\varphi_{\eta+\varsigma}^\alpha\big)}{\nu\big(\varphi_\eta^\alpha\big)} \exp\big(\alpha(A(\eta + \varsigma) - A(\eta))\big).$$

This can again be used to differentiate under the integral mappings such as $\eta \mapsto \mathbb{E}_{\check\varphi_\eta^\alpha}\Big[\prod_{j=1}^m S_{k_j}\Big]$ for $m \in \mathbb{N}$ and indices $k_1, \ldots, k_m \in \{1, \ldots, d\}$.

Consequently, the mappings $\eta \mapsto \nu\big(\varphi_\eta^\alpha\big)$ and $\mathcal{R} : \eta \mapsto \mathbb{E}_{\check\varphi_\eta^\alpha}[S]$ are of class $\mathcal{C}^\infty$ on $E$ with the gradient of the former being equal to

$$\partial_\eta \nu\big(\varphi_\eta^\alpha\big) = \alpha \int_Y \big(S(y) - \partial_\eta A(\eta)\big)\varphi_\eta^\alpha(y)\nu(\mathrm{d}y), \tag{22}$$

and the Jacobian matrix of the latter being given by

$$\mathbf{J}_\mathcal{R}(\eta) = \frac{1}{\nu(\varphi_\eta^\alpha)^2}\left[\nu(\varphi_\eta^\alpha)\left(\alpha\int_Y S(y)(S(y) - \mathbb{E}_{q_\eta}[S])^\top \varphi_\eta^\alpha(y)\nu(\mathrm{d}y)\right)\right.$$
$$\left. - \left(\int_Y S(y)\varphi_\eta^\alpha(y)\nu(\mathrm{d}y)\right)\left(\alpha\int_Y (S(y) - \mathbb{E}_{q_\eta}[S])^\top \varphi_\eta^\alpha(y)\nu(\mathrm{d}y)\right)\right],$$

which simplifies to

$$\mathbf{J}_\mathcal{R}(\eta) = \alpha\left[\int_Y S(y)(S(y) - \mathbb{E}_{q_\eta}[S])^\top \check\varphi_\eta^\alpha(y)\nu(\mathrm{d}y)\right.$$
$$\left. - \left(\int_Y S(y)\check\varphi_\eta^\alpha(y)\nu(\mathrm{d}y)\right)\left(\int_Y (S(y) - \mathbb{E}_{q_\eta}[S])^\top \check\varphi_\eta^\alpha(y)\nu(\mathrm{d}y)\right)\right]. \tag{23}$$

*Proof of Lemma 1.*  Let $\gamma \in (0, 1]$ and $\eta \in E$. By (12), the condition $\eta \in \mathrm{Fix}(\mathcal{M})$ is equivalent to $\partial A(\eta) = \gamma\mathcal{R}(\eta) + (1 - \gamma)\partial A(\eta)$. Since $\gamma > 0$, this equation simplifies to $\partial A(\eta) = \mathcal{R}(\eta)$, which concludes the proof since $\partial A(\eta) = \mathbb{E}_{q_\eta}[S]$ and, by definition, $\mathcal{R}(\eta) = \mathbb{E}_{\check\varphi_\eta^\alpha}[S]$. □

*Proof of Proposition 1.*  Using the properties of $\partial A$ and $\mathcal{R}$, the mapping $\mathcal{M}_\gamma$ defined in (12) is of class $\mathcal{C}^\infty$ on $E$. By applying formula (23) at $\eta_*$ and simplifying the expression, we get

$$\mathbf{J}_\mathcal{R}(\eta_\star) = \alpha\mathrm{Cov}_{\check\varphi_\star^\alpha}(S)\,. \tag{24}$$

Using the chain rule with (21), and once again the fact that $\eta_\star \in \text{Fix}(\mathcal{M})$, we obtain

$$\mathbf{J}_{\mathcal{M}_\gamma}(\eta_\star) = \gamma\alpha\text{Cov}_{q_\star}(S)^{-1}\text{Cov}_{\check{\varphi}_\star^\alpha}(S) + (1-\gamma)\text{Id}.$$

Recall that $\text{Cov}_{q_\star}(S)$ is positive definite. Multiplying by $\text{Cov}_{q_\star}(S)^{1/2}$ on the left and by $\text{Cov}_{q_\star}(S)^{-1/2}$ on the right shows that $\alpha\text{Cov}_{q_\star}(S)^{-1}\text{Cov}_{\check{\varphi}_\star^\alpha}(S)$ has the same complex eigenvalues as the positive semidefinite matrix $\alpha\text{Cov}_{q_\star}(S)^{-1/2}\text{Cov}_{\check{\varphi}_\star^\alpha}(S)\text{Cov}_{q_\star}(S)^{-1/2}$. Hence these eigenvalues are non-negative and we have

$$\begin{aligned}\varrho\left(\mathbf{J}_{\mathcal{M}_\gamma}(\eta_\star)\right) &= \varrho\left(\gamma\alpha\text{Cov}_{q_\star}(S)^{-1}\text{Cov}_{\check{\varphi}_\star^\alpha}(S)\right) + (1-\gamma)\\ &= 1 - \gamma\left(1 - \rho_\star\right).\end{aligned}$$

We also deduce that the Jacobian matrix $\mathbf{J}_{\mathcal{M}_\gamma}(\eta_\star)$ admits an eigenvector $x_\star \in \mathbb{R}^d$ for the eigenvalue $1 - \gamma\left(1 - \rho_\star\right)$. For any norm $\|\cdot\|_\bullet$ on $\mathbb{R}^d$, there is

$$\lim_{\varepsilon\to 0}\frac{\|\mathcal{M}_\gamma(\eta_\star + \varepsilon x_\star) - \mathcal{M}_\gamma(\eta_\star)\|_\bullet}{\varepsilon\|x_\star\|_\bullet} = \frac{\|\mathbf{J}_{\mathcal{M}_\gamma}(\eta_\star)x_\star\|_\bullet}{\|x_\star\|_\bullet} = 1 - \gamma\left(1 - \rho_\star\right),$$

which proves the inequality in (15). Moreover, for all $x \in \mathbb{R}^d$, there is

$$\left\|\mathbf{J}_{\mathcal{M}_\gamma}(\eta_\star)x\right\|_\star = \left\|\left(\gamma\alpha\text{Cov}_{q_\star}(S)^{-1/2}\text{Cov}_{\check{\varphi}_\star^\alpha}(S)\text{Cov}_{q_\star}(S)^{-1/2} + (1-\gamma)\text{Id}\right)\text{Cov}_{q_\star}(S)^{1/2}x\right\|.$$

Denoting by $\|\cdot\|_{\text{op}\star}$ and $\|\cdot\|_{\text{op}}$ the operator norms associated to $\|\cdot\|_\star$ and $\|\cdot\|$, respectively, we get

$$\begin{aligned}\left\|\mathbf{J}_{\mathcal{M}_\gamma}(\eta_\star)\right\|_{\text{op}\star} &= \sup_{x:\|\text{Cov}_{q_\star}(S)^{1/2}x\|\leq 1}\left\|\mathbf{J}_{\mathcal{M}_\gamma}(\eta_\star)x\right\|_\star\\ &= \left\|\gamma\alpha\text{Cov}_{q_\star}(S)^{-1/2}\text{Cov}_{\check{\varphi}_\star^\alpha}(S)\text{Cov}_{q_\star}(S)^{-1/2} + (1-\gamma)\text{Id}\right\|_{\text{op}}\\ &= 1 - \gamma\left(1 - \rho_\star\right),\end{aligned}$$

which proves the equality in (15).

Observe that the equivalence *(i)* ⟺ *(ii)* is a direct consequence of (15). Thus, it only remains to prove that *(i)* ⟹ *(iii)*. The gradient of the function $\mathcal{L}_\alpha$ is null at any fixed point of $\mathcal{M}_\gamma$. Indeed we can rewrite (16) as

$$\partial_\eta\mathcal{L}_\alpha(\eta) = \frac{\nu(\varphi_\eta^\alpha)}{\alpha - 1}\left(\mathbb{E}_{\check{\varphi}_\eta^\alpha}[S] - \mathbb{E}_{q_\eta}[S]\right),$$

which is zero at $\eta = \eta_\star$ by Lemma 1. Therefore, we only need to show that *(i)* is equivalent to the positive-definiteness of the Hessian matrix of $\mathcal{L}_\alpha$ at $\eta_\star$. We have, differentiating the above formula and using (21) and (22) at $\eta = \eta_\star$ along with (24),

$$\mathbf{H}_{\mathcal{L}_\alpha}(\eta_\star) = \frac{\nu(\varphi_\star^\alpha)}{\alpha - 1}\left(\alpha\text{Cov}_{\check{\varphi}_\star^\alpha}(S) - \text{Cov}_{q_\star}(S)\right).$$

Multiplying by $\text{Cov}_{q_\star}(S)^{-1/2}$ on both sides, $\mathbf{H}_{\mathcal{L}_\alpha}(\eta_\star)$ is positive definite if and only if the (non-negative) eigenvalues of $\alpha\text{Cov}_{q_\star}(S)^{-1/2}\text{Cov}_{\check{\varphi}_\star^\alpha}(S)\text{Cov}_{q_\star}(S)^{-1/2}$ are all less than 1. This condition is equivalent to *(i)*, as we have already proven that the largest eigenvalue of this matrix is $\rho_\star$. ☐

## B.1 Proof of Theorem 1

The proof of the theorem is based on the following lemma, inspired by [13, Theorem 6]. We do not apply the main result of this paper directly, as it requires the sequence to converge to some known value. We circumvent this difficulty by using the Banach fixed point theorem, a natural idea in light of the insight provided in Proposition 1.

**Lemma 2.** *Let $E$ be an open subset of $\mathbb{R}^d$ and $\mathcal{M} : E \to E$ be a continuous mapping. Starting from $\eta_0 \in E$, we construct the sequence $(\eta_t)$ by the recursion $\eta_{t+1} = \mathcal{M}(\eta_t)$. We assume that the following three conditions hold.*

    *(i) There exists a compact subset $\mathsf{K}$ of $E$ such that $\{\eta_t, t \geq 0\} \subset \mathsf{K}$.*

    *(ii) There exists a continuous function $\vartheta : E \to \mathbb{R}$ such that for all $\eta \in E$, $\vartheta \circ \mathcal{M}(\eta) \leq \vartheta(\eta)$, with equality if and only if $\eta = \mathcal{M}(\eta)$.*

    *(iii) For all $\eta \in \mathrm{Fix}(\mathcal{M})$, there exists $\rho'_\eta \in (0,1)$ such that, for all $\rho \in (\rho'_\eta, 1)$, we can find an open neighborhood $\mathcal{V}$ of $\eta$ on which the mapping $\mathcal{M}$ is a $\rho$-contraction.*

*Then the sequence $(\eta_t)$ converges to some $\eta_\star \in \mathrm{Fix}(\mathcal{M})$. Additionally, for all $\rho \in (\rho'_\star, 1)$, we have $\|\eta_t - \eta_\star\| = \mathcal{O}(\rho^t)$.*

*Proof.* There are two parts in this proof. First, we show that under *(i)* and *(ii)*, the sequence $(\eta_t)$ admits a limit point that is a fixed point of the mapping $\mathcal{M}$. Then, assumption *(iii)* allows us to apply the Banach fixed point theorem to conclude.

Since $\mathsf{K}$ in *(i)* is a compact set, there exist a strictly increasing function $\pi : \mathbb{N} \to \mathbb{N}$ and a point $\eta_\star \in \mathsf{K}$ such that $\eta_{\pi(t)} \longrightarrow \eta_\star$. Now, consider the sequence $(\eta_{\pi(t)+1})$. Once again, by compactness, there exist a strictly increasing function $\chi : \mathbb{N} \to \mathbb{N}$ and a point $\eta_{\star\star} \in \mathsf{K}$ such that $\eta_{\pi \circ \chi(t)+1} \longrightarrow \eta_{\star\star}$. Note that for all $t \geq 0$, we have

$$\pi \circ \chi(t+1) \geq \pi \circ \chi(t) + 1 \geq \pi \circ \chi(t),$$

and under *(ii)* this implies $\vartheta(\eta_{\pi \circ \chi(t+1)}) \leq \vartheta(\eta_{\pi \circ \chi(t)+1}) \leq \vartheta(\eta_{\pi \circ \chi(t)})$. By continuity of $\vartheta$, taking the limit and applying the squeeze theorem, we obtain $\vartheta(\eta_\star) = \vartheta(\eta_{\star\star})$. The sequence $(\eta_t)$ is defined such that $\mathcal{M}(\eta_{\pi \circ \chi(t)}) = \eta_{\pi \circ \chi(t)+1}$. The mapping $\mathcal{M}$ being continuous, taking $t \longrightarrow +\infty$ yields $\mathcal{M}(\eta_\star) = \eta_{\star\star}$. Applying $\vartheta$ on both sides of the previous identity yields $\vartheta \circ \mathcal{M}(\eta_\star) = \vartheta(\eta_{\star\star}) = \vartheta(\eta_\star)$. The equality case in *(ii)* implies $\mathcal{M}(\eta_\star) = \eta_\star$, i.e., the sequence $(\eta_t)$ has a limit point $\eta_\star$ that is a fixed point of the mapping $\mathcal{M}$.

Since $\eta_\star \in \mathrm{Fix}(\mathcal{M})$, assumption *(iii)* ensures the existence of a norm $\|\cdot\|_\star$ on $\mathbb{R}^d$ and of a neighborhood $\mathcal{V}$ of $\eta_\star$ such that $\mathcal{M}$ is $\rho$-Lipschitz continuous on $\mathcal{V}$ for the norm $\|\cdot\|_\star$, for all $\rho \in (\rho'_\star, 1)$. Recall that, by convention, $\rho'_\star$ shorthands for $\rho'_{\eta_\star}$. Thus there exists a non-empty open $\|\cdot\|_\star$-ball $B$ centered at $\eta_\star$ such that $\mathcal{M}$ is a $\rho$-contraction from $B$ to itself. Since $\eta_\star$ is a limit point of the sequence $(\eta_t)$, this sequence eventually falls into $B$. By the Banach fixed point theorem, the sequence $(\eta_t)_{t \geq t_0}$ converges to $\eta_\star$ at a geometric rate, namely $\|\eta_t - \eta_\star\| = \mathcal{O}(\rho^n)$. $\qquad\square$

Let us recall the monotonicity property that justifies algorithm (3) and its exact expression (6). This result is simply a reformulation of [9, Corollary 1] in a less general context.

**Theorem 4** (Monotonicity). *Let $\alpha \in (0,1)$, $\gamma \in (0,1]$, and define the mapping $\mathcal{M}_\gamma$ as in (12). Then, for all $\eta \in E$, if $\mathcal{M}_\gamma(\eta) \in E$, we have*

$$\mathcal{L}_\alpha \circ \mathcal{M}_\gamma(\eta) \leq \mathcal{L}_\alpha(\eta),$$

*with equality if and only if $\eta = \mathcal{M}_\gamma(\eta)$.*

We can now put these results together to prove Theorem 1.

*Proof of Theorem 1.* We apply Lemma 2 under the conditions (H1)–(H3). Condition *(i)* of Lemma 2 directly follows from (H2), and assumption (H1) guarantees that the mapping $\mathcal{M}_\gamma$ is continuous. By Theorem 4, we get condition *(ii)* with $\vartheta = \mathcal{L}_\alpha$ in Lemma 2. Finally, condition *(iii)* of Lemma 2 is verified by applying Proposition 1 with Assumption (H3), and since $\gamma_t \geq \delta$ eventually by (H4), we can take $\rho'_\star = 1 - \delta(1 - \rho_\star)$ in Lemma 2. $\qquad\square$

## B.2 Proof of Theorem 2

We start by stating a convergence result in a deterministic setting, which is a particular case of [11, Theorem 2].

**Lemma 3.** *Let $(\mu_t)$ be a sequence constructed by the recursion $\mu_{t+1} = \mu_t + \gamma_t \mathbf{M}_t \left[ h(\mu_t) + r_{t+1} \right]$ where the real- and vector-valued sequences $(\gamma_t)$ and $(r_t)$ verify $\sum \gamma_t = +\infty$, $\sum \gamma_t^2 < +\infty$, and $(\mathbf{M}_t)$ are positive definite matrices such that for all $t \geq 1$, $\mathrm{Sp}(\mathbf{M}_t) \subset [\lambda_{\min}, \lambda_{\max}]$, where $\lambda_{\max} > \lambda_{\min} > 0$, and $\mathrm{Sp}(\mathbf{M}_t)$ is the spectrum of $\mathbf{M}_t$. We also assume that the series $\sum \gamma_t \mathbf{M}_t r_{t+1}$ converges in $\mathbb{R}^d$. Further assume that $h$ is a continuous function, that there exists $\mathcal{L} : F \to \mathbb{R}$ satisfying $\partial \mathcal{L} = -h$, and that the points in $\mathcal{S} = \{\mu \in F, \ h(\mu) = 0\}$ are isolated. If the iterates in $(\mu_t)$ eventually belong to some compact $\mathsf{M} \subset F$ such that $\mathcal{S} \cap \mathsf{M} \neq \emptyset$, then $(\mu_t)$ converges to some point in $\mathcal{S}$.*

The idea is now to prove that the assumptions made in Lemma 3 hold almost surely. In what follows, we use the notation

$$G(\mu) = \hat{\mathcal{E}}(\mu; K) - \mu \hat{\ell}_\alpha^*(\mu; K), \tag{25}$$

where $\hat{\mathcal{E}}$ and $\hat{\ell}_\alpha^*$ are defined in (7) and (8)

**Lemma 4.** *Let $K$ be a positive integer. Using the notation introduced in (25), if (C2) holds, then*

$$\sup_{\mu \in \mathsf{M}} \mathbb{E}\left[ \|G(\mu)\|^2 \right] < +\infty. \tag{26}$$

*Proof.* By Minkowski's inequality, it suffices to prove the result for $K = 1$. In this case, setting $\eta = \partial A^{-1}(\mu)$, there is

$$\mathbb{E}\left[ \|G(\mu)\|^2 \right] = \mathbb{E}_\eta \left[ \|S - \mu\|^2 \left( \frac{p}{q_\eta} \right)^{2(1-\alpha)} \right].$$

The result follows immediately. $\square$

*Proof of Theorem 2.* The sequence $(\mu_t)$ is constructed by the recursion

$$\mu_{t+1} = \mu_t + \gamma_t \left[ \hat{\mathcal{E}}(\mu_t) - \mu_t \hat{\ell}_\alpha^*(\mu_t) \right] = \mu_t + \gamma_t \mathbf{M}_t \left[ h(\mu_t) + r_{t+1} \right],$$

introducing the matrix $\mathbf{M}_t = \mathbf{F}(\mu_t)$, the deterministic function $h(\mu) = \mathbf{F}^{-1}(\mu) \left[ \mathcal{E}(\mu) - \mu \ell_\alpha^*(\mu) \right]$ and the noise term $r_{t+1} = \mathbf{F}^{-1}(\mu_t) \left[ \hat{\mathcal{E}}(\mu_t) - \mu_t \hat{\ell}_\alpha^*(\mu_t) \right] - h(\mu_t)$. As explained in Section 4.1, we have $h = -\partial \mathcal{L}_\alpha^*$, and the zero set of $h$ is equal to the set $\mathcal{S}$ in assumption (C1). In particular, its points are isolated.

Let us now prove the existence of $\lambda_{\max} > \lambda_{\min} > 0$ such that $\mathrm{Sp}(\mathbf{M}_t) \subset [\lambda_{\min}, \lambda_{\max}]$ for all $t \geq 1$. Since we are working on $\Xi(\mathsf{M})$, we will assume without loss of generality that $\mu_t \in \mathsf{M}$ for all $t$. It thus suffices to show that the eigenvalues of $\mathbf{F}(\mu)$ can be bounded for all $\mu \in \mathsf{M}$. We have $\mathsf{M} \subset F$, and $\mathbf{F}(\mu)$ is a positive definite matrix for all $\mu \in F$. The application $\mu \mapsto \min \mathrm{Sp}(\mathbf{F}(\mu))$ is continuous, thence the Weierstrass extreme value theorem ensures that it attains a minimum on $\mathsf{M}$, and by the aforementioned positive-definiteness argument, this minimum cannot be 0. We denote it $\lambda_{\min}$. Similarly, $\lambda_{\max}$ can be defined as the maximum of the continuous function $\mu \mapsto \max \mathrm{Sp}(\mathbf{F}(\mu))$ on $\mathsf{M}$.

To apply Lemma 3, the only assumption left to check is the almost sure convergence of the series $\sum_t \gamma_t \mathbf{M}_t r_{t+1}$. The general term of this series is given by

$$\gamma_t \mathbf{F}(\mu_t) r_{t+1} = \gamma_t \left[ \hat{\mathcal{E}}(\mu_t) - \mu_t \hat{\ell}_\alpha^*(\mu_t) - \mathcal{E}(\mu_t) + \mu_t \ell_\alpha^*(\mu_t) \right],$$

and as explained in Section 4.1, it is a martingale increment sequence. To conclude, we only need to show that the martingale sequence $\sum_{t=0}^{n-1} \gamma_t \mathbf{M}_t r_{t+1}$ is bounded in $\mathcal{L}^2$. This is equivalent to showing $\sum_t \mathbb{E}\left[ \|\gamma_t \mathbf{M}_t r_{t+1}\|^2 \right] < +\infty$. Using (C2) and Lemma 4, we have

$$\begin{aligned} \mathbb{E}\left[ \|\gamma_t \mathbf{M}_t r_{t+1}\|^2 \right] &= \gamma_t^2 \, \mathbb{E}\left[ \mathbb{1}\{\mu_t \in \mathsf{M}\} \, \mathbb{E}\left[ \|r_{t+1}''\|^2 \right] \right] \\ &\leq \gamma_t^2 \sup_{\mu \in \mathsf{M}} \mathbb{E}\left[ \|G(\mu) - \mathbf{F}(\mu) h(\mu)\|^2 \right] \\ &\leq \gamma_t^2 \, M. \end{aligned}$$

By assumption (C0), the associated series converges. Lemma 3 allows us to conclude. $\square$

### B.3 Proof of Theorem 3

Theorem 3 is simply an application of [30, Theorem 1]. We recall a simplified version that fits the context of our analysis.

**Theorem 5.** *Let $\mu_\star \in F$ and $(\mu_t)$ be a sequence constructed by the stochastic algorithm*

$$\mu_{t+1} = \mu_t + \gamma_t\big[h(\mu_t) + r_{t+1}\big]$$

*with $(\gamma_t)$ a positive sequence, $h$ a deterministic function, and $(r_t)$ a random noise sequence. Let $(\mathcal{F}_t)$ be a filtration such that $(\mu_t)$ is $\mathcal{F}_t$-adapted, and assume*

   *(i) There exist $a > 1$ and a neighborhood $\mathcal{U}$ of $\mu_\star$ such that for all $\mu \in \mathcal{U}$, we may write $h(\mu) = H_\star(\mu - \mu_\star) + \mathcal{O}(\|\mu - \mu_\star\|^a)$, where $H_\star$ is a stable matrix.*

   *(ii) There exists $\delta \in (\frac{1}{2}, 1]$ such that for all $t \geq 1$, we have $\gamma_t = \gamma_0 t^{-\delta}$, with $\gamma_0 > 0$ if $\delta \in (\frac{1}{2}, 1)$, or $\gamma_0 > \frac{\beta}{2L(H_\star)}$ for some arbitrary $\beta \in (0, 1]$ if $\delta = 1$.*

   *(iii) There exist $R > 0$ and $b > 2/\delta$ such that $\mathbb{E}[r_{t+1} \,|\, \mathcal{F}_t]\mathbb{1}\{\|\mu_t - \mu_\star\| \leq R\} = 0$, and $\sup_{t \geq 0} \mathbb{E}\big[\|r_{t+1}\|^b \,|\, \mathcal{F}_t\big]\mathbb{1}\{\|\mu_t - \mu_\star\| \leq R\} < +\infty$.*

   *(iv) There exists a deterministic symmetric positive definite matrix $C$ such that $\lim \mathbb{E}\big[r_{t+1}r_{t+1}^\top \,|\, \mathcal{F}_t\big] = C$.*

*Then, letting $S_t = \sum_{i=1}^t \gamma_i$, there exists a real constant $\Lambda$ such that, almost surely on $\Gamma(\mu_\star)$, we have*

$$\limsup \sqrt{\frac{t}{\ln(S_t)}}\|\mu_t - \mu_\star\| \leq \Lambda.$$

*Proof of Theorem 3.* The proof consists in checking the assumptions of Theorem 5. First, the filtration $(\mathcal{F}_t)$ can be chosen as follows. We set $\mathcal{F}_0 = \sigma(\{\mu_0\})$ and $\mathcal{F}_t$ the $\sigma$-field generated by the (possibly random) initial parameter $\mu_0$ and the successive $(y_i)$ generated to compute $\hat{\mathcal{E}}(\mu_s)$ and $\hat{\ell}_\alpha^*(\mu_s)$ for $s = 1, \ldots, t$.

Condition *(ii)* on the design choice corresponds to (C0'), and *(iii)* is implied by (C2') using similar arguments as in Lemma 4 and that $h$ is continuous over $F$. Thus, we only need to check *(i)* and *(iv)*. Recall that in this proof, we have $h(\mu) = \mathcal{E}(\mu) - \mu\ell_\alpha^*(\mu)$ and $r_{t+1} = \hat{\mathcal{E}}(\mu_t) - \mu_t\hat{\ell}_\alpha^*(\mu_t) - h(\mu_t)$.

   *(i)* Rewriting (16), and using the properties of exponential families, for all $\mu \in F$,
   $$\partial_\mu h(\mu) = \partial_\mu\left[(\alpha - 1)\partial_\eta \mathcal{L}_\alpha^*(\mu)\right] = (\alpha - 1)\mathbf{F}^{-1}(\mu)\mathbf{H}_{\mathcal{L}_\alpha^*}(\mu).$$
   Since $\mu_\star \in \mathcal{S}$, we have $h(\mu_\star) = 0$. In other words, $\mathcal{E}(\mu_\star)/\ell_\alpha^*(\mu_\star) = \mu_\star$, which means that $\eta_\star = (\partial A)^{-1}(\mu_\star)$ is a fixed point of the mapping $\mathcal{M}_1$ defined in (12). By Proposition 1, the matrix $\mathbf{H}_{\mathcal{L}_\alpha}(\eta_\star)$ is positive definite. Using $\mathbf{H}_{\mathcal{L}_\alpha}(\eta) = \mathbf{H}_{\mathcal{L}_\alpha^*}(\mu)$ and the positive-definiteness of $\mathbf{F}^{-1}(\mu_\star)$ under (H1), we deduce that $H_\star = (\alpha - 1)\mathbf{F}^{-1}(\mu_\star)\mathbf{H}_{\mathcal{L}_\alpha^*}(\mu_\star)$ is a stable matrix. We conclude by applying Taylor's formula to the mapping $h$.

   *(iv)* Note that, for all $t$, $r_{t+1} = G(\mu_t) - \mathbb{E}[G(\mu_t)]$ where $G$ is defined as in Lemma 4. We deduce that $\mathbb{E}\big[r_{t+1}r_{t+1}^\top \,|\, \mathcal{F}_t\big]$ is equal to the matrix $\mathrm{Cov}\,(G(\mu))$ taken at (the random vector) $\mu = \mu_t$. To obtain *(iv)*, it thus suffices to show that $\mu \mapsto \mathrm{Cov}\,(G(\mu))$ is continuous on $F$ and valued in the set of positive definite matrices.

   Let $\eta \in E$ and $\mu = \partial A(\eta)$. We have
   $$\mathrm{Cov}\big(G(\mu)\big) = \frac{1}{K}\,\mathrm{Cov}_{q_\eta}\left((S - \mu)\left(\frac{p}{q_\eta}\right)^{1-\alpha}\right).$$

   This covariance matrix is positive definite since the exponential family is assumed to be minimal under Assumption (H1), which means that $S$ does not belong to an affine hyperplane under $q_\eta$. The continuity also follows from the preliminaries of the appendix.

   $\square$

## C    Additional results on the convergence of (11)

The following results are follow-ups on Theorem 3, they are all proven in [30], and reminded here for completeness.

**Theorem 6** (Law of the iterated logarithm, extended version). *Under the assumptions of Theorem 3, almost surely on $\Gamma(\mu_*)$,*

$$\limsup \sqrt{\frac{t}{\ln(S_t)}} \|\mu_t - \mu_*\| \leq \Lambda,$$

*where $\Lambda$ is a real constant and $S_t = \sum_{s=1}^{t} \gamma_s$. Moreover, if $w$ is an eigenvector of $H_*$, the Jacobian matrix of the function $h : \mu \mapsto \mathcal{E}(\mu) - \mu \ell_\alpha^*(\mu)$ taken at $\mu_*$, then*

$$\limsup \sqrt{\frac{t}{\ln(S_t)}} \langle w, \mu_t - \mu_* \rangle = -\liminf \sqrt{\frac{t}{\ln(S_t)}} \langle w, \mu_t - \mu_* \rangle = \sqrt{2 w^\top \Sigma w},$$

*where $\Sigma$ is the solution of the Lyapunov equation*

$$(H_* + \zeta \mathrm{Id})\Sigma + \Sigma(H_*^\top + \zeta \mathrm{Id}) = -D,$$

*with $\zeta = 0$ under (C0)-(i) and $\zeta = \beta/(2\gamma_0) < L(H_*)$ under (C0)-(ii).*

*Remark.*     *(i)* If $H_*$ is diagonalizable, we have $\Lambda = \|(P^\dagger)^{-1}\| \sqrt{2\mathrm{tr}(P^\dagger \Sigma P)}$, where $P$ a matrix which columns are eigenvectors of $H_*$, and $P^\dagger$ is its transconjugate.

 *(ii)* $\Sigma$ is given by the formula

$$\Sigma = \int_0^{+\infty} \exp\left[s(H_* + \zeta \mathrm{Id})\right] D \exp\left[s(H_*^\top + \zeta \mathrm{Id})\right] \mathrm{d}s.$$

**Theorem 7** (Quadratic strong law of large numbers). *Under the assumptions of Theorem 3, and using the same notations, almost surely on $\Gamma(\mu_*)$,*

$$\lim \frac{1}{S_t} \sum_{i=1}^{t} (\mu_i - \mu_*)(\mu_i - \mu_*)^\top = \Sigma.$$

**Theorem 8** (Central limit theorem). *Under the assumptions of Theorem 3, given $\Gamma(\mu_*)$,*

$$\sqrt{\gamma_t^{-1}} (\mu_t - \mu_*) \xrightarrow{(d)} \mathcal{N}(0, \Sigma).$$

# D Additional figures

## D.1 Toy experiment with a Gaussian mixture target

In the case of a Gaussian mixture target, approximated by a single Gaussian density, we find that the initial point and the learning rate have a high influence on whether the algorithms converge, where they converge, and at what speed.

In most cases, we find that UNB is at least as good SGM, SGE and NAT, while being less computationally intensive than SGM and NAT, and more stable than SGE. An improper tuning of $\gamma_0$ may cause some procedures to diverge, SGE being by far the most sensitive, followed by NAT and SGM. Divergence very rarely happens for MAX, which requires little to no tuning to converge. Another noticeable effect is that MAX matches the unbiased procedures in terms of value of the alpha-divergence if $\gamma_t$ is too large and there are enough samples.

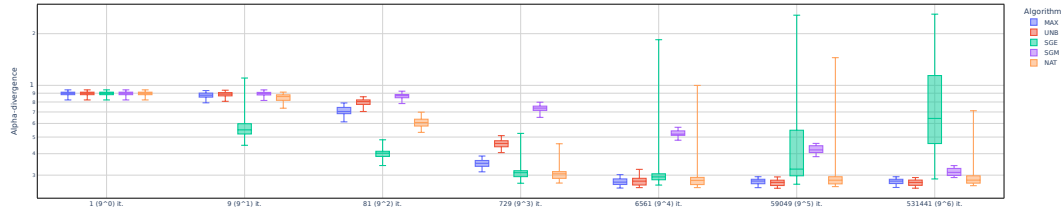

Figure 3: Boxplot of the alpha-divergence after set numbers of iterations with large step sizes. Approximating a Gaussian mixture with a single Gaussian.

In the case of the Gaussian approximation of a Cauchy target, Assumption (C2) is not verified. However, we still observe convergence for all algorithms in most cases. As in the Gaussian mixture example, SGM appears to be the slowest method, while SGE sometimes diverges due to an unfavorable loss landscape.

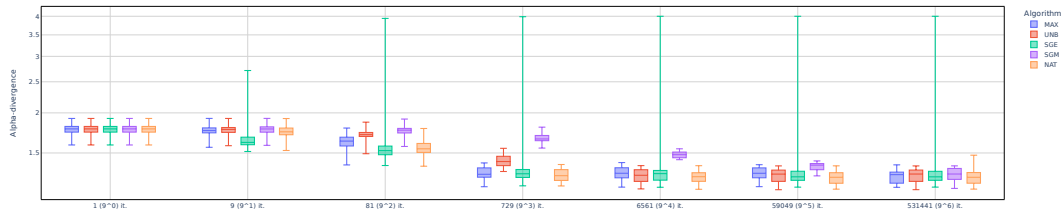

Figure 4: Boxplot of the alpha-divergence after set numbers of iterations. Approximating a Cauchy density with a single Gaussian.

